# A Method for Learning from Hints

**Yaser S. Abu-Mostafa**
Departments of Electrical Engineering, Computer Science,
and Computation and Neural Systems
California Institute of Technology
Pasadena, CA 91125
e-mail: yaser@caltech.edu

## Abstract

We address the problem of learning an unknown function by putting together several pieces of information (hints) that we know about the function. We introduce a method that generalizes learning from examples to learning from hints. A canonical representation of hints is defined and illustrated for new types of hints. All the hints are represented to the learning process by examples, and examples of the function are treated on equal footing with the rest of the hints. During learning, examples from different hints are selected for processing according to a given schedule. We present two types of schedules; fixed schedules that specify the relative emphasis of each hint, and adaptive schedules that are based on how well each hint has been learned so far. Our learning method is compatible with any descent technique that we may choose to use.

## 1 INTRODUCTION

The use of hints is coming to the surface in a number of research communities dealing with learning and adaptive systems. In the learning-from-examples paradigm, one often has access not only to examples of the function, but also to a number of hints (prior knowledge, or side information) about the function. The most common difficulty in taking advantage of these hints is that they are heterogeneous and cannot be easily integrated into the learning process. This paper is written with the specific goal of addressing this problem. The paper develops a systematic method

for incorporating different hints in the usual learning-from-examples process.

Without such a systematic method, one can still take advantage of certain types of hints. For instance, one can implement an invariance hint by preprocessing the input to achieve the invariance through normalization. Alternatively, one can structure the learning model in a way that directly implements the invariance (Minsky and Papert, 1969). Whenever direct implementation is feasible, the full benefit of the hint is realized. This paper does not attempt to offer a superior alternative to direct implementation. *However, when direct implementation is not an option, we prescribe a systematic method for incorporating practically any hint in any descent technique for learning.* The goal is to automate the use of hints in learning to a degree where we can effectively utilize a large number of different hints that may be available in a practical situation. As the use of hints becomes routine, we are encouraged to exploit even the simplest observations that we may have about the function we are trying to learn.

The notion of hints is quite general and it is worthwhile to formalize what we mean by a hint as far as our method is concerned. Let $f$ be the function that we are trying to learn. A hint is a property that $f$ is known to have. Thus, all that is needed to qualify as a hint is to have a litmus test that $f$ passes and that can be applied to different functions. Formally, a hint is a given subset of functions that includes $f$.

We start by introducing the basic nomenclature and notation. The *environment* $X$ is the set on which the function $f$ is defined. The points in the environment are distributed according to some probability distribution $P$. $f$ takes on values from some set $Y$

$$f : X \rightarrow Y$$

Often, $Y$ is just $\{0, 1\}$ or the interval $[0, 1]$. The *learning process* takes pieces of information about (the otherwise unknown) $f$ as input and produces a *hypothesis* $g$

$$g : X \rightarrow Y$$

that attempts to approximate $f$. The degree to which a hypothesis $g$ is considered an approximation of $f$ is measured by a distance or 'error'

$$E(g, f)$$

The error $E$ is based on the disagreement between $g$ and $f$ as seen through the eyes of the probability distribution $P$.

Two popular forms of the error measure are

$$E = \Pr[g(x) \neq f(x)]$$

and

$$E = \mathcal{E}[(g(x) - f(x))^2]$$

where $\Pr[.]$ denotes the probability of an event, and $\mathcal{E}[.]$ denotes the expected value of a random variable. The underlying probability distribution is $P$. $E$ will always be a non-negative quantity, and we will take $E(g, f) = 0$ to mean that $g$ and $f$ are identical for all intents and purposes. We will also assume that when the set of hypotheses is parameterized by real-valued parameters (e.g., the weights in the case of a neural network), $E$ will be well-behaved as a function of the parameters

(in order to allow for derivative-based descent techniques). We make the same assumptions about the error measures that will be introduced in section 2 for the hints.

In this paper, the 'pieces of information' about $f$ that are input to the learning process are more general than in the learning-from-examples paradigm. In that paradigm, a number of points $x_1, \cdots, x_N$ are picked from $X$ (usually independently according to the probability distribution $P$) and the values of $f$ on these points are provided. Thus, the input to the learning process is the set of examples

$$(x_1, f(x_1)), \cdots, (x_N, f(x_N))$$

and these examples are used to guide the search for a good hypothesis. We will consider the set of examples of $f$ as only one of the available hints and denote it by $H_0$. The other hints $H_1, \cdots, H_M$ will be additional known facts about $f$, such as invariance properties for instance.

The paper is organized as follows. Section 2 develops a canonical way for representing different hints. This is the first step in dealing with any hint that we encounter in a practical situation. Section 3 develops the basis for learning from hints and describes our method, including specific learning schedules.

## 2   REPRESENTATION OF HINTS

As we discussed before, a hint $H_m$ is defined by a litmus test that $f$ satisfies and that can be applied to the set of hypotheses. This definition of $H_m$ can be extended to a definition of 'approximation of $H_m$' in several ways. For instance, $g$ can be considered to approximate $H_m$ within $\epsilon$ if there is a function $h$ that strictly satisfies $H_m$ for which $E(g, h) \leq \epsilon$. In the context of learning, it is essential to have a notion of approximation since exact learning is seldom achievable. Our definitions for approximating different hints will be part of the scheme for representing those hints.

The first step in representing $H_m$ is to choose a way of generating 'examples' of the hint. For illustration, suppose that $H_m$ asserts that

$$f : [-1, +1] \rightarrow [-1, +1]$$

is an *odd* function. An example of $H_m$ would have the form

$$f(-x) = -f(x)$$

for a particular $x \in [-1, +1]$. To generate $N$ examples of this hint, we generate $x_1, \cdots, x_N$ and assert for each $x_n$ that $f(-x_n) = -f(x_n)$. Suppose that we are in the middle of a learning process, and that the current hypothesis is $g$ when the example $f(-x) = -f(x)$ is presented. We wish to measure how much $g$ disagrees with this example. This leads to the second component of the representation, the error measure $e_m$. For the oddness hint, $e_m$ can be defined as

$$e_m = (g(x) + g(-x))^2$$

so that $e_m = 0$ reflects total agreement with the example (i.e., $g(-x) = -g(x)$). Once the disagreement between $g$ and an example of $H_m$ has been quantified

through $e_m$, the disagreement between $g$ and $H_m$ as a whole is automatically quantified through $E_m$, where

$$E_m = \mathcal{E}(e_m)$$

The expected value is taken w.r.t. the probability rule for picking the examples. Therefore, $E_m$ can be estimated by averaging $e_m$ over a number of examples that are independently picked.

The choice of representation of $H_m$ is not unique, and $E_m$ will depend on the form of examples, the probability rule for picking the examples, and the error measure $e_m$. A minimum requirement on $E_m$ is that it should be zero when $E = 0$. This requirement guarantees that a hypothesis for which $E = 0$ (perfect hypothesis) will not be excluded by the condition $E_m = 0$.

Let us illustrate how to represent different types of hints. Perhaps the most common type of hint is **the invariance hint**. This hint asserts that $f(x) = f(x')$ for certain pairs $x, x'$. For instance, "$f$ is shift-invariant" is formalized by the pairs $x, x'$ that are shifted versions of each other. To represent the invariance hint, an invariant pair $(x, x')$ is picked as an example. The error associated with this example is

$$e_m = (g(x) - g(x'))^2$$

Another related type of hint is **the monotonicity hint** (or inequality hint). The hint asserts for certain pairs $x, x'$ that $f(x) \leq f(x')$. For instance, "$f$ is monotonically nondecreasing in $x$" is formalized by all pairs $x, x'$ such that $x < x'$. To represent the monotonicity hint, an example $(x, x')$ is picked, and the error associated with this example is given by

$$e_m = \begin{cases} (g(x) - g(x'))^2 & \text{if } g(x) > g(x') \\ 0 & \text{if } g(x) \leq g(x') \end{cases}$$

The third type of hint we discuss here is **the approximation hint**. The hint asserts for certain points $x \in X$ that $f(x) \in [a_x, b_x]$. In other words, the value of $f$ at $x$ is known only approximately. The error associated with an example $x$ of the approximation hint is

$$e_m = \begin{cases} (g(x) - a_x)^2 & \text{if } g(x) < a_x \\ (g(x) - b_x)^2 & \text{if } g(x) > b_x \\ 0 & \text{if } g(x) \in [a_x, b_x] \end{cases}$$

Another type of hints arises when the learning model allows non-binary values for $g$ where $f$ itself is known to be binary. This gives rise to **the binary hint**. Let $\hat{X} \subseteq X$ be the set where $f$ is known to be binary (for Boolean functions, $\hat{X}$ is the set of binary input vectors). The binary hint is represented by examples of the form $x$, where $x \in \hat{X}$. The error function associated with an example $x$ (assuming 0/1 binary convention, and assuming $g(x) \in [0, 1]$) is

$$e_m = g(x)(1 - g(x))$$

This choice of $e_m$ forces it to be zero when $g(x)$ is either 0 or 1, while it would be positive if $g(x)$ is between 0 and 1.

It is worth noting that the set of examples of $f$ can be formally treated as a hint, too. Given $(x_1, f(x_1)), \cdots, (x_N, f(x_N))$, **the examples hint** asserts that these are the correct values of $f$ at those particular points. Now, to generate an 'example' of this hint, we pick a number $n$ from 1 to $N$ and use the corresponding $(x_n, f(x_n))$. The error associated with this example is $e_0$ (we fix the convention that $m = 0$ for the examples hint)

$$e_0 = (g(x_n) - f(x_n))^2$$

Assuming that the probability rule for picking $n$ is uniform over $\{1, \cdots, N\}$,

$$E_0 = \mathcal{E}(e_0) = \frac{1}{N} \sum_{n=1}^{N} (g(x_n) - f(x_n))^2$$

In this case, $E_0$ is also the best estimator of $E = \mathcal{E}[(g(x) - f(x))^2]$ given $x_1, \cdots, x_N$ that are independently picked according to the original probability distribution $P$. This way of looking at the examples of $f$ justifies their treatment exactly as one of the hints, and underlines the distinction between $E$ and $E_0$.

In a practical situation, we try to infer as many hints about $f$ as the situation will allow. Next, we represent each hint according to the scheme discussed in this section. This leads to a list $H_0, H_1, \cdots, H_M$ of hints that are ready to produce examples upon the request of the learning algorithm. We now address how the algorithm should pick and choose between these examples as it moves along.

## 3   LEARNING SCHEDULES

If the learning algorithm had complete information about $f$, it would search for a hypothesis $g$ for which $E(g, f) = 0$. However, $f$ being unknown means that the point $E = 0$ cannot be directly identified. The most any learning algorithm can do given the hints $H_0, H_1, \cdots, H_M$ is to reach a hypothesis $g$ for which all the error measures $E_0, E_1, \cdots, E_M$ are zeros. Indeed, we have required that $E = 0$ implies that $E_m = 0$ for all $m$.

If that point is reached, regardless of how it is reached, the job is done. However, it is seldom the case that we can reach the zero-error point because either (1) it does not exist (i.e., no hypothesis can satisfy all the hints simultaneously, which implies that no hypothesis can replicate $f$ exactly), or (2) it is difficult to reach (i.e., the computing resources do not allow us to exhaustively search the space of hypotheses looking for that point). In either case, we will have to settle for a point where the $E_m$'s are 'as small as possible'.

How small should each $E_m$ be? A balance has to be struck, otherwise some $E_m$'s may become very small at the expense of the others. This situation would mean that some hints are over-learned while the others are under-learned. We will discuss learning schedules that use different criteria for balancing between the hints. The schedules are used by the learning algorithm to simultaneously minimize the $E_m$'s. Let us start by exploring how simultaneous minimization of a number of quantities is done in general.

Perhaps the most common approach is that of *penalty functions* (Wismer and Chat-

tergy, 1978). In order to minimize $E_0, E_1, \cdots, E_M$, we minimize the penalty function

$$\sum_{m=0}^{M} \alpha_m E_m$$

where each $\alpha_m$ is a non-negative number that may be constant (exact penalty function) or variable (sequential penalty function). Any descent technique can be employed to minimize the penalty function once the $\alpha_m$'s are selected. The $\alpha_m$'s are weights that reflect the relative emphasis or 'importance' of the corresponding $E_m$'s. The choice of the weights is usually crucial to the quality of the solution.

Even if the $\alpha_m$'s are determined, we still do not have the explicit values of the $E_m$'s in our case (recall that $E_m$ is the expected value of the error $e_m$ on an example of the hint). Instead, we will estimate $E_m$ by drawing several examples and averaging their error. Suppose that we draw $N_m$ examples of $H_m$. The estimate for $E_m$ would then be

$$\frac{1}{N_m} \sum_{n=1}^{N_m} e_m^{(n)}$$

where $e_m^{(n)}$ is the error on the $n^{\text{th}}$ example. Consider a batch of examples consisting of $N_0$ examples of $H_0$, $N_1$ examples of $H_1$, $\cdots$ , and $N_M$ examples of $H_M$. The total error of this batch is

$$\sum_{m=0}^{M} \sum_{n=1}^{N_m} e_m^{(n)}$$

If we take $N_m \propto \alpha_m$, this total error will be a proportional estimate of the penalty function

$$\sum_{m=0}^{M} \alpha_m E_m$$

In effect, we translated the weights into a **schedule**, where different hints are emphasized, not by magnifying their error, but by representing them with more examples.

A batch of examples can be either a *uniform batch* that consist of $N$ examples of one hint at a time, or, more generally, a *mixed batch* where examples of different hints are allowed within the same batch. If the descent technique is linear and the learning rate is small, a schedule that uses mixed batches is equivalent to a schedule that alternates between uniform batches (with frequency equal to the frequency of examples in the mixed batch). If we are using a nonlinear descent technique, it is generally more difficult to ascertain a direct translation from mixed batches to uniform batches, but there may be compelling heuristic correspondences. All schedules discussed here are expressed in terms of uniform batches for simplicity.

The implementation of a given schedule goes as follows: (1) The algorithm decides which hint (which $m$ for $m = 0, 1, \cdots, M$) to work on next, according to some criterion; (2) The algorithm then requests a batch of examples of this hint; (3) It performs its descent on this batch; and (4) When it is done, it goes back to step (1). We make a distinction between *fixed schedules*, where the criterion for selecting the hint can be 'evaluated' ahead of time (albeit time-invariant or time-varying,

deterministic or stochastic), and *adaptive schedules,* where the criterion depends on what happens as the algorithm runs. Here are some fixed and adaptive schedules:

**Simple Rotation:** This is the simplest possible schedule that tries to balance between the hints. It is a fixed schedule that rotates between $H_0, H_1, \cdots, H_M$. Thus, at step $k$, a batch of $N$ examples of $H_m$ is processed, where $m = k \bmod (M + 1)$. This simple-minded algorithm tends to do well in situations where the $E_m$'s are somewhat similar.

**Weighted Rotation:** This is the next step in fixed schedules that tries to give different emphasis to different $E_m$'s. The schedule rotates between the hints, visiting $H_m$ with frequency $\nu_m$. The choice of the $\nu_m$'s can achieve balance by emphasizing the hints that are more important or harder to learn.

**Maximum Error:** This is the simplest adaptive schedule that tries to achieve the same type of balance as simple rotation. At each step $k$, the algorithm processes the hint with the largest error $E_m$. The algorithm uses estimates of the $E_m$'s to make its selection.

**Maximum Weighted Error:** This is the adaptive counterpart to weighted rotation. It selects the hint with the largest value of $\nu_m E_m$. The choice of the $\nu_m$'s can achieve balance by making up for disparities between the numerical ranges of the $E_m$'s. Again, the algorithm uses estimates of the $E_m$'s.

Adaptive schedules attempt to answer the question: Given a set of values for the $E_m$'s, which hint is the most under-learned? The above schedules answer the question by comparing the individual $E_m$'s. Although this works well in simple cases, it does not take into consideration the correlation between different hints. As we deal with more and more hints, the correlation between the $E_m$'s becomes more significant. This leads us to the final schedule that achieves the balance between the $E_m$'s through their relation to the actual error $E$.

**Adaptive Minimization:** Given the estimates of $E_0, E_1, \cdots, E_M$, make $M + 1$ estimates of $E$, each based on all but one of the hints:

$$\hat{E}(\bullet, E_1, E_2, \cdots, E_M)$$
$$\hat{E}(E_0, \bullet, E_2, \cdots, E_M)$$
$$\hat{E}(E_0, E_1, \bullet, \cdots, E_M)$$
$$\cdots$$
$$\hat{E}(E_0, E_1, E_2, \cdots, \bullet)$$

and choose the hint for which the corresponding estimate is the *smallest.*

In other words, $E$ becomes the common thread between the $E_m$'s. Knowing that we are really trying to minimize $E$, and that the $E_m$'s are merely a vehicle to this end, *the criterion for balancing the $E_m$'s should be based on what is happening to $E$ as far as we can tell.*

# CONCLUSION

This paper developed a systematic method for using different hints as input to the learning process, generalizing the case of invariance hints (Abu-Mostafa, 1990). The method treats all hints on equal footing, including the examples of the function. Hints are represented in a canonical way that is compatible with the common learning-from-examples paradigm. No restrictions are made on the learning model or the descent technique to be used.

The hints are captured by the error measures $E_0, E_1, \cdots, E_M$, and the learning algorithm attempts to simultaneously minimize these quantities. The simultaneous minimization of the $E_m$'s gives rise to the idea of balancing between the different hints. A number of algorithms that minimize the $E_m$'s while maintaining this balance were discussed in the paper. Adaptive schedules in particular are worth noting because they automatically compensate against many artifacts of the learning process.

It is worthwhile to distinguish between the quality of the hints and the quality of the learning algorithm that uses these hints. The quality of the hints is determined by how reliably one can predict that the actual error $E$ will be close to zero for a given hypothesis based on the fact that $E_0, E_1, \cdots, E_M$ are close to zero for that hypothesis. The quality of the algorithm is determined by how likely it is that the $E_m$'s will become nearly as small as they can be within a reasonable time.

### Acknowledgements

The author would like to thank Ms. Zehra Kök for her valuable input. This work was supported by the AFOSR under grant number F49620-92-J-0398.

### References

Abu-Mostafa, Y. S. (1990), Learning from hints in neural networks, *Journal of Complexity* **6**, 192-198.

Al-Mashouq, K. and Reed, I. (1991), Including hints in training neural networks, *Neural Computation* **3**, 418-427.

Minsky, M. L. and Papert, S. A. (1969), "Perceptrons," MIT Press.

Omlin, C. and Giles, C. L. (1992), Training second-order recurrent neural networks using hints, *Machine Learning: Proceedings of the Ninth International Conference (ML-92)*, D. Sleeman and P. Edwards (ed.), Morgan Kaufmann.

Suddarth, S. and Holden, A. (1991), Symbolic neural systems and the use of hints for developing complex systems, *International Journal of Machine Studies* **35**, p. 291.

Wismer, D. A. and Chattergy, R. (1978), "Introduction to Nonlinear Optimization," North Holland.